# Learning to Find Pictures of People

**Sergey Ioffe**
Computer Science Division
U.C. Berkeley
Berkeley CA 94720
*ioffe@cs.berkeley.edu*

**David Forsyth**
Computer Science Division
U.C. Berkeley
Berkeley CA 94720
*daf@cs.berkeley.edu*

## Abstract

Finding articulated objects, like people, in pictures presents a particularly difficult object recognition problem. We show how to find people by finding putative body segments, and then constructing assemblies of those segments that are consistent with the constraints on the appearance of a person that result from kinematic properties. Since a reasonable model of a person requires at least nine segments, it is not possible to present every group to a classifier. Instead, the search can be pruned by using projected versions of a classifier that accepts groups corresponding to people. We describe an efficient projection algorithm for one popular classifier, and demonstrate that our approach can be used to determine whether images of real scenes contain people.

## 1 Introduction

Several typical collections containing over ten million images are listed in [2]. There is an extensive literature on obtaining images from large collections using features computed from the whole image, including colour histograms, texture measures and shape measures; a partial review appears in [5].

However, in the most comprehensive field study of usage practices (a paper by Enser [2] surveying the use of the Hulton Deutsch collection), there is a clear user preference for searching these collections on image semantics. An ideal search tool would be a quite general object recognition system that could be adapted quickly and easily to the types of objects sought by a user. An important special case is finding people and determining what they are doing. This is hard, because people have many internal degrees of freedom. We follow the approach of [3], and represent people as collections of cylinders, each representing a body segment. Regions that could be the projections of cylinders are easily found using techniques similar to those of [1]. Once these regions are found, they must be assembled

into collections that are consistent with the appearance of images of real people, which are constrained by the kinematics of human joints; consistency is tested with a classifier. Since there are many candidate segments, a brute force search is impossible. We show how this search can be pruned using projections of the classifier.

## 2   Learning to Build Segment Configurations

Suppose that $N$ segments have been found in an image, and there are $m$ body parts. We will define a *labeling* as a set $L = \{(l_1, s_1), (l_2, s_2), \ldots, (l_k, s_k)\}$ of pairs where each segment $s_i \in \{1 \ldots N\}$ is labeled with the *label* $l_i \in \{1 \ldots m\}$. A labeling is *complete* if it represents a full $m$-segment configuration (Fig. 2(a,b)).

Assume we have a classifier $C$ that for any complete labeling $L$ outputs $C(L) > 0$ if $L$ corresponds to a person-like configuration, and $C(L) < 0$ otherwise. Finding all the possible body configurations in an image is equivalent to finding all the complete labelings $L$ for which $C(L) > 0$. This cannot be done with brute-force search through the entire set. The search can be pruned if, for an (incomplete) labeling $L'$ there is no complete $L \supseteq L'$ such that $C(L) > 0$. For instance, if two segments cannot represent the upper and lower left arm, as in Figure 1a, then we do not consider any complete labelings where they are labeled as such.

*Projected classifiers* make the search for body configurations efficient by pruning labelings using the properties of smaller sub-labelings (as in [7], who use manually determined bounds and do not learn the tests). Given a classifier $C$ which is a function of a set of features whose values depend on segments with labels $l_1 \ldots l_m$, the *projected classifier* $C_{l_1 \ldots l_k}$ is a function of of all those features that depend only on the segments with labels $l_1 \ldots l_k$. In particular, $C_{l_1 \ldots l_k}(L') > 0$ if there is some extension $L$ of $L'$ such that $C(L) > 0$ (see figure 1). The converse need not be true: the feature values required to bring a projected point inside the positive volume of $C$ may not be realized with any labeling of the current set of segments $1, \ldots, N$. For a projected classifier to be useful, it must be easy to compute the projection, and it must be effective in rejecting labelings at an early stage. These are strong requirements which are not satisfied by most good classifiers; for example, in our experience a support vector machine with a positive definite quadratic kernel projects easily but typically yields unrestrictive projected classifiers.

### 2.1   Building Labelings Incrementally

Assume we have a classifier $C$ that accepts assemblies corresponding to people and that we can construct projected classifiers as we need them. We will now show how to use them to construct labelings, using a *pyramid of classifiers*.

A pyramid of classifiers (Fig. 1(c)), determined by the classifier $C$ and a permutation of labels $(l_1 \ldots l_k)$ consists of nodes $N_{l_i \ldots l_j}$ corresponding to each of the projected classifiers $C_{l_i \ldots l_j}$, $i \leq j$. Each of the bottom-level nodes $N_{l_i}$ receives the set of all segments in the image as the input. The top node $N_{l_1 \ldots l_m}$ outputs the set of all complete labelings $L = \{(l_1, s_1) \ldots (l_m, s_m)\}$ such that $C(L) > 0$, i.e. the set of all assemblies in the image classified as people. Further, each node $N_{l_i \ldots l_j}$ outputs the set of all sub-labelings $L = \{(l_i, s_i) \ldots (l_j, s_j)\}$ such that $C_{l_i \ldots l_j}(L) > 0$.

The nodes $N_{l_i}$ at the bottom level work by selecting all segments $s_i$ in the image for which $C_{l_i}\{(l_i, s_i)\} > 0$. Each of the remaining nodes has two parts: merging and filtering. The *merging* stage of node $N_{l_i \ldots l_j}$ merges the outputs of its children by computing the set of all labelings $\{(l_i, s_i) \ldots (l_j, s_j)\}$ where $\{(l_i, s_i) \ldots (l_{j-1}, s_{j-1})\}$

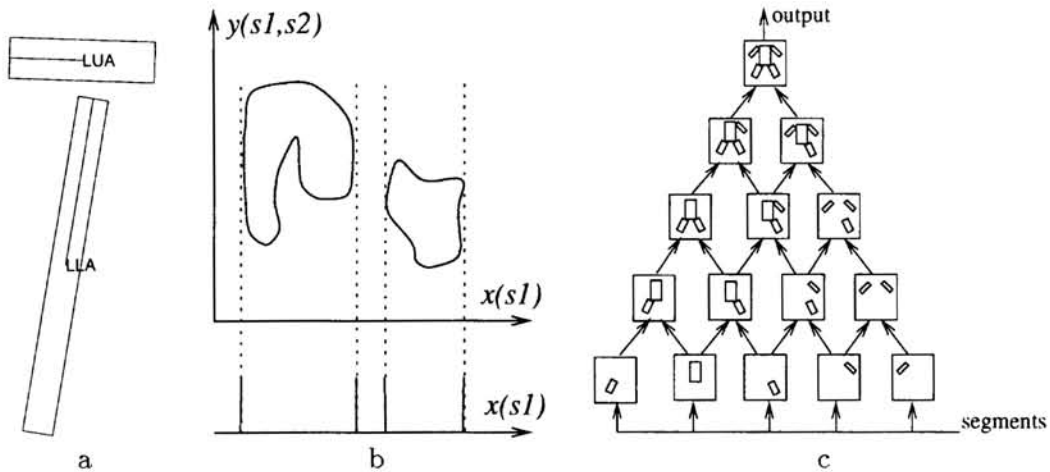

Figure 1: (a) *Two segments that cannot correspond to the left upper and lower arm. Any configuration where they do can be rejected using a projected classifier regardless of the other segments that might appear in the configuration.* (b) *Projecting a classifier* $C\{(l_1, s_1), (l_2, s_2)\}$. *The shaded area is the volume classified as positive, for the feature set* $\{x(s_1), y(s_1, s_2)\}$. *Finding the projection* $C_{l_1}$ *amounts to projecting off the features that cannot be computed from* $s_1$ *only, i.e.,* $y(s_1, s_2)$. (c) *A pyramid of classifiers. Each node outputs sub-assemblies accepted by the corresponding projected classifier. Each node except those in the bottom row works by forming labelings from the outputs of its two children, and filtering the result using the corresponding projected classifier. The top node outputs the set of all complete labelings that correspond to body configurations.*

and $\{(l_{i+1}, s_{i+1}) \dots (l_j, s_j)\}$ are in the outputs of $N_{l_i \dots l_{j-1}}$ and $N_{l_{i+1} \dots l_j}$, respectively. The *filtering* stage then selects, from the resulting set of labelings, those for which $C_{l_i \dots l_j}(\cdot) > 0$, and the resulting set is the output of $N_{l_i \dots l_j}$. It is clear, from the definition of projected classifiers, that the output of the pyramid is, in fact, the set of all complete $L$ for which $C(L) > 0$ (note that $C_{l_1 \dots l_m} = C$).

The only constraint on the order in which the outputs of nodes are computed is that children nodes have to be applied before parents. In our implementation, we use nodes $N_{l_i \dots l_j}$ where $j$ changes from 1 to $m$, and, for each $j$, $i$ changes from $j$ down to 1. This is equivalent to computing sets of labelings of the form $\{(l_1, s_1) \dots (l_j, s_j)\}$ in order, where getting $(j + 1)$-segment labelings from $j$-segment ones is itself an incremental process, whereby we check labels against $l_{j+1}$ in the order $l_j, l_{j-1}, \dots, l_1$. In practice, we choose the latter order on the fly for each increment step using a greedy algorithm, to minimize the size of labeling sets that are constructed (note that in this case the classifiers no longer form a pyramid). The order $(l_1 \dots l_m)$ in which labels are added to an assembly needs to be fixed. We determine this order with a greedy algorithm by running a large segment set through the labeling builder and choosing the next label to add so as to minimize the number of labelings that result.

## 2.2   Classifiers that Project

In our problem, each *segment* from the set $\{1 \dots N\}$ is a rectangle in some position and orientation. Given a complete labeling $L = \{(1, s_1), \dots, (m, s_m)\}$, we want to have $C(L) > 0$ iff the segment arrangement produced by $L$ looks like a person.

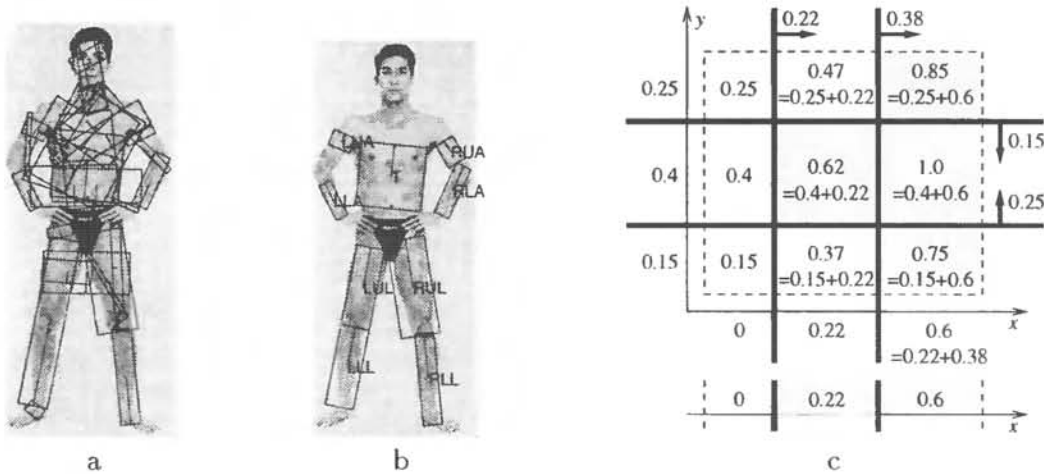

Figure 2: (a) *All segments extracted for an image.* (b) *A labeled segment con-figuration corresponding to a person, where* T=*torso,* LUA=*left upper arm, etc. The head is not marked because we are not looking for it with our method. The single left leg segment in (a) has been broken in (b) to generate the upper and lower leg segments.* (c) *(top) A combination of a bounding box (the dashed line) and a boosted classifier, for two features x and y. Each plane in the boosted classifier is a thick line with the positive half-space indicated by an arrow; the associated weight β is shown next to the arrow. The shaded area is the positive volume of the classifier, which are the points P where* $\sum_f w_f(P(f)) > 1/2$. *The weights $w_x(\cdot)$ and $w_y(\cdot)$ are shown along the x- and y-axes, respectively, and the total weight $w_x(P(x)) + w_y(P(y))$ is shown for each region of the bounding box. (bottom) The projected classifier, given by $w_x(P(x)) > 1/2 - \delta = 0.1$ where $\delta = \max_{P(y)} w_y(P(y)) = \max\{0.25, 0.4, 0.15\} = 0.4$.*

Each feature will depend on a few segments (1 to 3 in our experiments). Our kinematic features are invariant to translation, uniform scaling or rotation of the segment set, and include angles between segments and ratios of lengths, widths and distances. We expect the features that correspond to human configurations to lie within small fractions of their possible value ranges. This suggests using an axis-aligned bounding box, with bounds learned from a collection of positive labelings, for a good first separation, and then using a boosted version of a weak classifier that splits the feature space on a single feature value (as in [6]). This classifier projects particularly well, using a simple algorithm described in section 2.3.

Each weak classifier (Fig. 2(c)) is defined by the feature $f_j$ on which the split is made, the position $p_j$ of the splitting hyperplane, and the direction $d_j \in \{1, -1\}$ that determines which half-space is positive. A point $P$ is classified as positive iff $d_j(P(f_j) - p_j) > 0$, where $P(f_j)$ is the value of feature $f_j$. The boosting algorithm will associate a weight $\beta_j$ with each plane (so that $\sum_j \beta_j = 1$), and the resulting classifier will classify a point as positive iff $\sum_{d_j(P(f_j)-p_j)>0} \beta_j > 1/2$, that is, iff the total weight of the weak classifiers that classify the point as positive is at least a half of the total weight of the classifiers. The set $\{f_j\}$ may have repeating features (which may have different $p_j, d_j$ and $w_j$ values), and does not need to span the entire feature set.

By grouping together the weights corresponding to planes splitting on the same feature, we finally rewrite the classifier as $\sum_f w_f(P(f)) > 1/2$, where $w_f(P(f)) =$

$\sum_{f_j=f,\ d_j(P(f)-p_j)>0} \beta_j$ is the weight associated with the particular value of feature $f$, is a piece-wise constant function and depends on in which of the intervals given by $\{p_j|f_j=f\}$ this value falls.

## 2.3   Projecting a Boosted Classifier

Given a classifier constructed as above, we need to construct classifiers that depend on on some identified subset of the features. The geometry of our classifiers — whose positive regions consist of unions of axis-aligned bounding boxes — makes this easy to do.

Let $g$ be the feature to be projected away — perhaps because the value depends on a label that is not available. The projection of the classifier should classify a point $P'$ in the (lower-dimensional) feature space as positive iff $\max_P \sum_f w_f(P(f)) > 1/2$ where $P$ is a point which projects into $P'$ but can have any value for $P(g)$. We can rewrite this expression as $\sum_{f\neq g} w_f(P'(f)) + \max_{P(g)} w_g(P(g)) > 1/2$. The value of $\delta = \max w_g(P(g))$ is readily available and independent of $P'$. We can see that, with the feature projected away, we obtain $\sum_f w_f(P'(f)) > 1/2 - \delta$. Any number of features can be projected away in a sequence in this fashion. An example of the projected classifier is shown in Figure 2(c).

The classifier $C$ we are using allows for an efficient building of labelings, in that the features do not need to be recomputed when we move from $C_{l_1...l_k}$ to $C_{l_1..l_{k+1}}$. We achieve this efficiency by carrying along with a labeling $L = \{(l_1,s_1)...(l_k,s_k)\}$ the sum $\sigma(L) = \sum_{f\in F(l_1...l_k)} w_f(P(f))$ where $F(l_1...l_k)$ is the set of all features computable from the segments labeled as $l_1,...,l_k$, and $\{P(f)\}$ — the values of these features. When we add another segment to get $L' = \{(l_1,s_1)...(l_{k+1},s_{k+1})\}$, we can compute $\sigma(L') = \sigma(L) + \sum_{f\in F(l_1...l_{k+1})\backslash F(l_1...l_k)} w_f(P'(f))$. In other words, when we add a label $l_{k+1}$, we need to compute only those features that require $s_{k+1}$ for their computation.

## 3   Experimental Results

We report results for a system that automatically identifies potential body segments (using the techniques described in [4]), and then applies the assembly process described above. Images for which assemblies that are kinematically consistent with a person are reported as having people in them. The segment finder may find either 1 or 2 segments for each limb, depending on whether it is bent or straight; because the pruning is so effective, we can allow segments to be broken into two equal halves lengthwise (like the left leg in Fig. 2(b)), both of which are tested.

### 3.1   Training

The training set included 79 images without people, selected randomly from the COREL database, and 274 images each with a single person on uniform background. The images with people have been scanned from books of human models [10]. All segments in the test images were reported; in the control images, only segments whose interior corresponded to human skin in colour and texture were reported. Control images, both for the training and for the test set, were chosen so that all had at least 30% of their pixels similar to human skin in colour and texture. This gives a more realistic test of the system performance by excluding regions that are obviously not human, and reduces the number of segments in the control images to the same order of magnitude as those in the test images.

| Features | Test | Control |
|:---:|:---:|:---:|
| *367* | 120 | 28 |
| *567* | 120 | 86 |

a

| Features | False Neg. | False Pos. |
|:---:|:---:|:---:|
| *367* | 37 % | 4 % |
| *567* | 49 % | 10 % |

b

Table 1: *(a) Number of images of people (test) and without people (control) processed by the classifiers with 367 and 567 features. (b) False negative (images with a person where no body configuration was found) and false positive (images with no people where a person was detected) rates.*

The models are all wearing either swim suits or no clothes, otherwise segment finding fails; it is an open problem to segment people wearing loose clothing. There is a wide variation in the poses of the training examples, although all body segments are visible. The sets of segments corresponding to people were then hand-labeled. Of the 274 images with people, segments for each body part were found in 193 images. The remaining 81 resulted in incomplete configurations, which could still be used for computing the bounding box used to obtain a first separation. Since we assume that if a configuration looks like a person then its mirror image would too, we double the number of body configurations by flipping each one about a vertical axis. The bounding box is then computed from the resulting 548 points in the feature space, without looking at the images without people.

The boosted classifier was trained to separate two classes: the $193 \times 2 = 386$ points corresponding to body configurations, and 60727 points that did not correspond to people but lay in the bounding box, obtained by using the bounding box classifier to incrementally build labelings for the images with no people. We added 1178 synthetic positive configurations obtained by randomly selecting each limb and the torso from one of the 386 real images of body configurations (which were rotated and scaled so the torso positions were the same in all of them) to give an effect of joining limbs and torsos from different images rather like children's flip-books. Remarkably, the boosted classifier classified each of the real data points correctly but misclassified 976 out of the 1178 synthetic configurations as negative; the synthetic examples were unexpectedly more similar to the negative examples than the real positive examples were.

## 3.2 Results

The test dataset was separate from the training set and included 120 images with a person on a uniform background, and varying numbers of control images, reported in Table 1. We report results for two classifiers, one using 567 features and the other using a subset of 367 of those features. Table 1b shows the false positive and false negative rates achieved for each of the two classifiers. By marking 51% of test images and only 10% of control images, the classifier using 567 features compares extremely favorably with that of [3], which marked 54% of test images and 38% of control images using hand-tuned tests to form groups of four segments. In 55 of the 59 images where there was a false negative, a segment corresponding to a body part was missed by the segment finder, meaning that the overall system performance significantly understates the classifier performance. There are few signs of overfitting, probably because the features are highly redundant. Using the larger set of features makes labeling faster (by a factor of about five), because more configurations are rejected earlier.

## 4  Conclusions and Future Work

Groups of segments that satisfy kinematic constraints, learned from images of real people, quite reliably correspond to people and can be used to identify them. Our trick of projecting classifiers is effective at pruning an otherwise completely unmanageable correspondence search. Future issues include: fusing responses from face finders (such as those of [11, 9]; exploiting patterns of shading on human limbs to get better selectivity (as in [8]); determining the configuration of the person, which might tell what they are doing; and exploiting the kinematic similarities between humans and many animals to build systems that can find many different types of animal without searching the classes one by one.

## References

[1] J.M. Brady and H. Asada. Smoothed local symmetries and their implementation. *International Journal of Robotics Research*, 3(3), 1984.

[2] P.G.B. Enser. Query analysis in a visual information retrieval context. *J. Document and Text Management*, 1(1):25–52, 1993.

[3] M. M. Fleck, D. A. Forsyth, and C. Bregler. Finding naked people. In *European Conference on Computer Vision 1996. Vol. II*, pages 592–602, 1996.

[4] D.A. Forsyth and M.M. Fleck. Body plans. In *IEEE Conf. on Computer Vision and Pattern Recognition*, 1997.

[5] D.A. Forsyth, J. Malik, M.M. Fleck, H. Greenspan, T. Leung, S. Belongie, C. Carson, and C. Bregler. Finding pictures of objects in large collections of images. In *Proc. 2'nd International Workshop on Object Representation in Computer Vision*, 1996.

[6] Y. Freund and R.E. Schapire. Experiments with a new boosting algorithm. In *Machine Learning - 13*, 1996.

[7] W.E.L. Grimson and T. Lozano-Pérez. Localizing overlapping parts by searching the interpretation tree. *IEEE Trans. Patt. Anal. Mach. Intell.*, 9(4):469–482, 1987.

[8] J. Haddon and D.A. Forsyth. Shading primitives. In *Int. Conf. on Computer Vision*, 1997. to appear.

[9] H.A. Rowley, S. Baluja, and T. Kanade. Human face detection in visual scenes. In D.S. Touretzky, M.C. Mozer, and M.E. Hasselmo, editors, *Advances in Neural Information Processing 8*, pages 875–881, 1996.

[10] Elte Shuppan. *Pose file*, volume 1-7. Books Nippan, 1993-1996. A collection of photographs of human models, annotated in Japanese.

[11] K-K Sung and T. Poggio. Example based learning for view based face detection. Ai memo 1521, MIT, 1994.
